# Persistent Homology for Learning Densities with Bounded Support

Florian T. Pokorny      Carl Henrik Ek      Hedvig Kjellström      Danica Kragic [*]
Computer Vision and Active Perception Lab, Centre for Autonomous Systems
School of Computer Science and Communication
KTH Royal Institute of Technology, Stockholm, Sweden
`{fpokorny, chek, hedvig, danik}@csc.kth.se`

## Abstract

We present a novel method for learning densities with bounded support which enables us to incorporate 'hard' topological constraints. In particular, we show how emerging techniques from computational algebraic topology and the notion of persistent homology can be combined with kernel-based methods from machine learning for the purpose of density estimation. The proposed formalism facilitates learning of models with bounded support in a principled way, and – by incorporating persistent homology techniques in our approach – we are able to encode algebraic-topological constraints which are not addressed in current state of the art probabilistic models. We study the behaviour of our method on two synthetic examples for various sample sizes and exemplify the benefits of the proposed approach on a real-world dataset by learning a motion model for a race car. We show how to learn a model which respects the underlying topological structure of the racetrack, constraining the trajectories of the car.

## 1   Introduction

Probabilistic methods based on Gaussian densities have celebrated successes throughout machine learning. They are the crucial ingredient in Gaussian mixture models (GMM) [1], Gaussian processes [2] and Gaussian mixture regression (GMR) [3] which have found applications in fields such as robotics, speech recognition and computer vision [1, 4, 5] to name just a few. While Gaussian distributions are convenient to work with for several theoretical and practical reasons (the central limit theorem, easy computation of means and marginals, etc.) they do fall into the class of densities on $\mathbb{R}^d$ for which $\operatorname{supp} f = \mathbb{R}^d$; i.e. *they assign a non-zero probability to every subset with non-zero volume in $\mathbb{R}^d$*. This property of Gaussians can be problematic if an application dictates that certain subsets of space should constitute a 'forbidden' region having zero probability mass. A simple example would be a probabilistic model of admissible positions of a robot in an indoor environment, where one wants to assign zero – rather than just 'low' – probability to positions corresponding to collisions with the environment. Encoding such constraints using *e.g.* a Gaussian mixture model is not natural since it assigns potentially low, but non-zero probability mass to every portion of space.

In contrast to the above Gaussian models, we consider non-parametric density estimators based on spherical kernels with bounded support. As we shall explain, this enables us to study topological properties of the support region $\Omega_\varepsilon$ for such estimators. Kernel-based density estimators are well-established in the statistical literature [6] with the basic idea being that one should put a rescaled version of a given model density over each observed data-point to obtain an estimate for the probability density from which the data was sampled. The choice of rescaling – or 'bandwidth' – $\varepsilon$ has been studied with respect to the standard $L^1$ and $L^2$ error and is still an active area of research [7]. We focus particularly on spherical truncated Gaussian kernels here which have been some-

---
[*]This work was supported by the EU projects FLEXBOT (FP7-ERC-279933) and TOMSY (IST-FP7-270436) and the Swedish Foundation for Strategic Research

what overlooked as a tool for probabilistic modelling. An important aspect of these kernels is that their associated conditional and marginal distributions can be computed analytically, enabling us to efficiently work with them in the context of probabilistic inference.

A different interpretation of a density with support in an $\varepsilon$-ball can be given using the notion of bounded noise. There, one assumes that observations are distorted by noise following a density with bounded support (instead of e.g. Gaussian noise). Bounded noise models are used in the signal processing community for robust filtering and estimation [8, 9], but to our knowledge, we are the first to combine densities with bounded support and topology to model the underlying structure of data. Thinking of a set of observations $S = \{X_1, ..., X_n\} \subset \mathbb{R}^n$ as 'fuzzy up to noise in an $\varepsilon$-ball' naturally leads one to consider the space $\Omega_\varepsilon(S) = \bigcup_i \mathbb{B}_\varepsilon(X_i)$ of balls of size $\varepsilon$ around the data points. Persistent homology is a novel tool for studying topological properties of spaces such as $\Omega_\varepsilon(S)$ which has emerged from the field of computational algebraic topology in recent years [10, 11]. Using persistent homology, it becomes possible to study clustering, periodicity and more generally the existence of 'holes' of various dimensions in $\Omega_\varepsilon(S)$ for $\varepsilon$ lying in an interval.

Starting from the basic observation that one can construct a kernel-based density estimator $\hat{f}_\varepsilon$ whose region of support is exactly $\Omega_\varepsilon(S)$, this paper investigates the interplay between the topological information contained in $\Omega_\varepsilon(S)$ and a corresponding density estimate. Specifically, we make the following contributions:

- Given prior topological information about supp $f = \Omega$, we define a topologically admissible bandwidth interval $[\varepsilon_{min}, \varepsilon_{max}]$ and propose and evaluate a topological bandwidth selector $\varepsilon_{top} \in [\varepsilon_{min}, \varepsilon_{max}]$.

- Given no prior topological information, we explain how persistent homology can be of use to determine a topologically admissible bandwidth interval.

- We describe how additional constraints defining a forbidden subset $F \subset \mathbb{R}^n$ of the parameter-space can be incorporated into our topological bandwidth estimation framework.

- We provide quantitative results on synthetic data in 1D and 2D evaluating the expected $L^2$ errors for density estimators with topologically chosen bandwidth values $\varepsilon \in \{\varepsilon_{min}, \varepsilon_{mid}, \varepsilon_{max}, \varepsilon_{top}\}$. We carry out this evaluation for various spherical kernels and compare our results to an asymptotically optimal bandwidth choice.

- We use our method in a learning by demonstration [12] context and compare our results with a current state of the art Gaussian mixture regression method.

## 2 Background

### 2.1 Kernel-based density estimation

Let $S = \{X_1, ..., X_n\} \subset \mathbb{R}^d$ be an i.i.d. sample arising from a probability density $f : \mathbb{R}^d \to \mathbb{R}$. *Kernel-based density estimation* [13, 14, 15] is an approach for reconstructing $f$ from the sample by means of an estimator $\hat{f}_{\varepsilon,n}(x) = \frac{1}{n\varepsilon^d} \sum_{i=1}^n K\left(\frac{x - X_i}{\varepsilon}\right)$, where the kernel function $K : \mathbb{R}^d \to \mathbb{R}$ is a suitably chosen probability density. In this context, $\varepsilon > 0$ is called the *bandwidth*. If one is only interested in an estimator that minimizes the expected $L^2$ norm of $\hat{f}_{\varepsilon,n} - f$, the choice of $\varepsilon$ is crucial, while the particular choice of kernel $K$ is generally less important [7, 6]. Let $\{\varepsilon_n\}_{n=1}^\infty$ be a sequence of positive bandwidth values depending on the sample size $n$. It follows from classical results [14, 15] that for any sufficiently well-behaved density $K$, $\lim_{n\to\infty} \mathbb{E}[(\hat{f}_{\varepsilon_n,n}(x) - f(x))^2] = 0$ provided that $\lim_{n\to\infty} \varepsilon_n = 0$ and $\lim_{n\to\infty} n\varepsilon_n^d = \infty$. Despite this encouraging result, the question of determining the best bandwidth for a given sample is an ongoing research topic and the interested reader is referred to the review [7] for an in-depth discussion. One branch of methods [6] tries to minimize the *Mean Integrated Squared Error*, $MISE(\varepsilon_n) = \mathbb{E}\left[\int (\hat{f}_{\varepsilon_n,n}(x) - f(x))^2 \mathrm{d}x\right]$.

An asymptotic analysis reveals that, under mild conditions on $K$ and $f$ [6], $MISE(\varepsilon_n)$ can be approximated asymptotically by $AMISE(\varepsilon_n)$ as $n \to \infty$ if $\lim_{n\to\infty} \varepsilon_n = 0$ and $\lim_{n\to\infty} n\varepsilon_n^d = \infty$. Here, AMISE denotes the *Asymptotic Mean Integrated Squared Error*. If we consider only spherical kernels that are symmetric functions of the norm $\|x\|$ of their input variable $x$, an asymptotic analysis [6] shows that, in dimension $d$,

$$AMISE(\varepsilon_n) = \frac{1}{n\varepsilon_n^d} \int K(x)^2 \, \mathrm{d}x + \frac{\varepsilon_n^4}{4} \mu_2(K)^2 \int \{\mathrm{tr}(\mathrm{Hess}\, f(x))\}^2 \, \mathrm{d}x,$$

where $\mu_2(K) = \int x_j^2 K(x) \mathrm{d}x$ is independent of the choice of $j \in \{1,\ldots,d\}$ by the spherical symmetry and $\mathrm{tr}(\mathrm{Hess}\, f(x))$ denotes the trace of the Hessian of $f$ at $x$. Due to the availability of a relatively simple explicit formula for AMISE, a large class of bandwidth selection methods attempt to estimate and minimize AMISE instead of working with MISE directly. One finds that AMISE is minimized for

$$\varepsilon_{amise}(n) = \left( \frac{1}{n} \frac{d \int K(x)^2 \,\mathrm{d}x}{\mu_2(K)^2 \int \left\{\mathrm{tr}(\mathrm{Hess}\, f(x))\right\}^2 \,\mathrm{d}x} \right)^{\frac{1}{4+d}}.$$

Since $f$ is assumed unknown in real world examples, so called plug-in methods can be used to approximate $\varepsilon_{amise}$ [7]. In this paper, we will work with two synthetic examples of densities for which we can compute $\varepsilon_{amise}$ numerically in order to benchmark our topological bandwidth selection procedure. For our experiments, we choose three spherical kernels $K : \mathbb{R}^d \to \mathbb{R}$ that are defined to be zero outside the unit ball $\mathbb{B}_1(0)$ and are defined by $K_u = \mathrm{Vol}(\mathbb{B}_1(0))^{-1}$ (uniform), $K_c(x) = \frac{d(d+1)\Gamma(\frac{d}{2})}{2\pi^{\frac{d}{2}}}(1-\|x\|)$ (conic) and $K_t(x) = (2\pi\sigma^2)^{-\frac{d}{2}} \left(1 - \frac{\Gamma\left(\frac{d}{2}, \frac{1}{2\sigma^2}\right)}{\Gamma\left(\frac{d}{2}\right)}\right)^{-1} e^{-\frac{\|x\|^2}{2\sigma^2}}$ (truncated Gaussian) respectively for $\|x\| \leqslant 1$. These kernels can be defined in any dimension $d > 0$ and are *spherical*, i.e. they are functions of the radial distance to the origin only which enables us to efficiently evaluate them and to sample from the corresponding estimator $\hat{f}_{\varepsilon,n}$ even when the dimension $d$ is very large. We will denote the standard spherical Gaussian by $K_e(x) = (2\pi\sigma^2)^{-\frac{d}{2}} e^{-\frac{\|x\|^2}{2\sigma^2}}$.

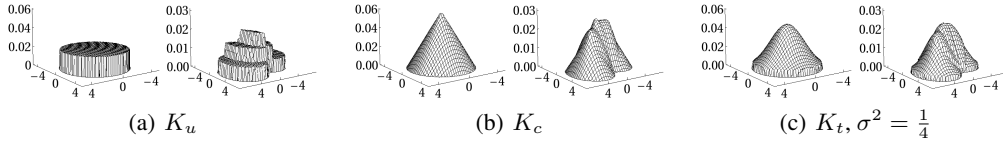

(a) $K_u$        (b) $K_c$        (c) $K_t, \sigma^2 = \frac{1}{4}$

Figure 1: $\frac{1}{4^2} K\left(\frac{x}{4}\right)$ for the indicated kernels and a corresponding estimator $\hat{f}_{4,3}$ for three sample points.

## 2.2 Persistent homology

Consider the point cloud $S$ shown in Figure 2(a). For a human observer, it is noticeable that $S$ looks 'circular'. One can reformulate the existence of the 'hole' in Figure 2(a) in a mathematically precise way using persistent homology [16] which has recently gained increasing traction as a tool for the analysis of structure in point-cloud data [10].

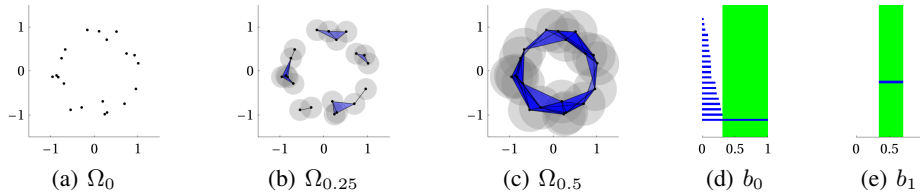

(a) $\Omega_0$     (b) $\Omega_{0.25}$     (c) $\Omega_{0.5}$     (d) $b_0$     (e) $b_1$

Figure 2: Noisy data concentrated around a circle (a) and corresponding barcodes in dimension zero (d) and one (e). In (b) and (c), we display $\Omega_\varepsilon$ for $\varepsilon = 0.25, 0.5$ respectively together with the corresponding Vietoris-Rips complex $\mathcal{V}_{2\varepsilon}$ which we use for approximating the topology of $\Omega_\varepsilon$. While the vertical axis in the $i^{th}$ barcode has no special meaning, the horizontal axis displays the $\varepsilon$ parameter of $\mathcal{V}_{2\varepsilon}$. At any fixed $\varepsilon$ value, the number of bars lying above and containing $\varepsilon$ is equal to the $i^{th}$ Betti number of $\mathcal{V}_{2\varepsilon}$. The shaded region highlights the $\varepsilon$-interval for which $\mathcal{V}_{2\varepsilon}$ has one connected component (i.e. $b_0(\mathcal{V}_{2\varepsilon}) = 1$) in (d) and for which a single 'circle' (i.e. $b_1(\mathcal{V}_{2\varepsilon}) = 1$) is detected in (e).

In the approach of [10], one starts with a subset $\Omega \subset \mathbb{R}^d$ and assumes that there exists some probability density $f$ on $\mathbb{R}^d$ that is concentrated near $\Omega$. Given an i.i.d. sample $S = \{X_1, \cdots, X_n\}$ from the corresponding probability distribution, one of the aims of persistent homology in this setting is to recover some of the topological structure of $\Omega$ – the homology groups $H_i(\Omega, \mathbb{Z}_2)$, for $i = 1, \ldots, d$ – from the sample $S$. Each $H_i(\Omega, \mathbb{Z}_2)$ is a vector space over $\mathbb{Z}_2$ and its dimension $b_i(\Omega)$ is called

the $i^{th}$ Betti number. One of the properties of homology is that homology groups are invariant under a large class of deformations (*i.e.* homotopies) of the underlying topological space. A popular example of such a deformation is to consider a teacup that is continuously deformed into a doughnut. One can think of $b_0(\Omega)$ as measuring the number of connected components while, roughly, $b_i(\Omega)$, for $i > 0$ describes the number $i$-dimensional holes of $\Omega$. A closed curve in $\mathbb{R}^d$ that does not self-intersect can for example be classified by $b_0 = 1$ (it has one connected component) and $b_1 = 1$ (it is topologically a circle). The reader is encouraged to consult [17] for a rigorous introduction to homotopies and related concepts.

Given a discrete sample $S$ and a distance parameter $\varepsilon > 0$, consider the set $\Omega_\varepsilon(S) = \bigcup_{i=1}^{n} \mathbb{B}_\varepsilon(X_i)$, for $\varepsilon \in [0, \infty)$, where $\mathbb{B}_\varepsilon(p) = \{x \in \mathbb{R}^d : \|x - p\| \leqslant \varepsilon\}$. In Figure 2(b) and 2(c) this set is displayed for increasing $\varepsilon$ values. $\Omega_\varepsilon(S)$ is a topological space and, in the case where $\Omega$ is a smooth compact submanifold in $\mathbb{R}^d$ and $f$ is in a very restrictive class of densities with support in a small tubular neighbourhood around $\Omega$, [18, 11] have proven results showing that $\Omega_\varepsilon(S)$ is homotopy equivalent to $\Omega$ with high probability for certain large sample sizes. The key insight of persistent homology is that we should study not just the homology of $\Omega_\varepsilon(S)$ for a fixed value of $\varepsilon$ but for all $\varepsilon \in [0, \infty)$ simultaneously. The idea is then to study how the homology groups $H_i(\Omega_\varepsilon(S), \mathbb{Z}_2)$ change with $\varepsilon$ and one records the changes in Betti number using a *barcode* [10] (see *e.g.* figure 2(d) and 2(e)). Computing the barcode corresponding to $H_i(\Omega_\varepsilon(S), \mathbb{Z}_2)$ directly (via the Čech complex given by our covering of balls $\mathbb{B}_\varepsilon(X_1), \ldots, \mathbb{B}_\varepsilon(X_n)$ [10]) is computationally very expensive and one hence computes the barcode corresponding to the homology groups of the Vietoris-Rips complex $\mathcal{V}_{2\varepsilon}(S)$. This complex is an abstract complex with vertices given by the elements of $S$ and where we insert a $k$-simplex for every set of $k + 1$ distinct elements of $S$ such that any two are within distance less than $2\varepsilon$ of each other (see [10]). The homology groups of $\mathcal{V}_{2\varepsilon}(S)$ are not necessarily isomorphic to the homology groups of $\Omega_\varepsilon(S)$, but can serve as an approximation due to the interleaving property of the Vietoris-Rips and Čech complex, see *e.g.* Prop 2.6 [10]. For the computation of barcodes, we use the javaPlex software [19]. The computed $i^{th}$ barcode then records the birth and death times of topological features of $\mathcal{V}_{2\varepsilon}$ in dimension $i$ as we increase $\varepsilon$ from zero to some maximal value $M$, where $M$ is called the *maximal filtration value*.

## 3 Our framework

Given a dataset $S = \{X_1, \ldots, X_n\} \subset \mathbb{R}^d$, sampled in an i.i.d. fashion from an underlying probability distribution with density $f : \mathbb{R}^d \to \mathbb{R}$ with bounded support $\Omega$, we propose to recover $f$ using a kernel density estimator $\hat{f}_{\varepsilon,n}$ in a way that respects the algebraic topology of $\Omega$. For this, we consider only $\hat{f}_{\varepsilon,n}$ based on kernels $K$ with $\operatorname{supp} K = \mathbb{B}_1(0)$, and in particular, we experiment with $K_t, K_u$ and $K_c$. For such kernels, $\operatorname{supp} \hat{f}_{\varepsilon,n} = \Omega_\varepsilon(S) = \cup_{i=1}^{n} \mathbb{B}_\varepsilon(X_i)$ whose topological features we can approximate by computing the barcodes for $\mathcal{V}_{2\varepsilon}$.

If no prior information on the topological features of $\Omega$ is given, we can then inspect these barcodes and search for large intervals in which the Betti numbers do not change. This approach is used in [10], who demonstrated that topological features of data can be discovered in this way. Alternatively, one might be given prior information on the Betti numbers (e.g. using knowledge of periodicity, number of clusters, inequalities involving Betti numbers) that one can incorporate by searching for $\varepsilon$-intervals on which such constraints are satisfied. Geometric constraints on the data can additionally be incorporated by restricting to allowable $\varepsilon$-intervals to values for which $\Omega_\varepsilon(S)$ does not contain 'forbidden regions'. In the robotics setting, frequently encountered examples for such forbidden regions are singular points in the joint space of a robot, or positions in space corresponding to collisions with the environment.

Let us now assume that we are given constraints on some of the Betti numbers of $\Omega$. For a given sample $S$, we then compute the barcodes for $\mathcal{V}_{2\varepsilon}$ in each dimension $i \in \{1, \ldots, d\}$ up to a large maximal value $M$ using javaPlex [19] and determine the set $A$ of admissible $\varepsilon$ values. If $A$ is empty, we consider the topological reconstruction to have failed. This will happen, for example, if our assumptions about the data are incorrect, or if we do not have enough samples to reconstruct $\Omega$. If $A$ is non-empty, we attempt to determine a finite union of disjoint intervals on which the Betti numbers constraints are satisfied. Since, in our experiments, the interval $I = [\varepsilon_{min}(n), \varepsilon_{max}(n)]$ (determined up to some fixed precision) with smallest possible $\varepsilon_{min}(n)$ among those coincided with the largest such interval in most cases (indicating stable topological features), we decided to

investigate this $I \subset A$ for further analysis. For $\varepsilon \in [\varepsilon_{min}(n), \varepsilon_{max}(n)]$, the resulting density $\hat{f}_{\varepsilon,n}$ then has a support region $\Omega_\varepsilon(S)$ with the correct Betti numbers – as approximated by $\mathcal{V}_{2\varepsilon}$. We note the following elementary observation:

**Lemma 3.1.** *Let* $d \in \mathbb{N}$ *and* $\varepsilon_{min}(n), \varepsilon_{max}(n) \in \mathbb{R}$ *for all* $n \in \mathbb{N}$. *Suppose that* $\lim_{n\to\infty} \varepsilon_{min}(n) = 0$ *and that there exists* $a, b \in \mathbb{R}$ *such that* $0 < a < \varepsilon_{max}(n) < b$ *and* $0 \leqslant \varepsilon_{min}(n) < \varepsilon_{max}(n)$ *for all* $n \in \mathbb{N}$. *Then* $\varepsilon_{top}(n) = \varepsilon_{min}(n) + \frac{\varepsilon_{max}(n) - \varepsilon_{min}(n)}{2} n^{-\frac{1}{4+d}}$ *satisfies i)* $\varepsilon_{top}(1) = \varepsilon_{mid}(1)$ *and* $\varepsilon_{top}(n) \in [\varepsilon_{min}(n), \varepsilon_{mid}(n)]$ *for all* $n \in \mathbb{N}$, *where we define* $\varepsilon_{mid}(n) = \frac{\varepsilon_{max}(n) + \varepsilon_{min}(n)}{2}$ *ii)* $\lim_{n\to\infty} \varepsilon_{top}(n) = 0$ *and iii)* $\lim_{n\to\infty} n\varepsilon_{top}(n)^d = \infty$.

It is our intuition that, for a large class of constraints on the Betti numbers and for tame densities $f : \mathbb{R}^d \to \mathbb{R}$ (such as densities concentrated on a neighbourhood of a compact submanifold of $\mathbb{R}^d$ [11]), $\varepsilon_{min}(n)$ and $\varepsilon_{max}(n)$ exist for all large enough sample sizes $n$ with high probability and that the conditions of Lemma 3.1 are satisfied. In that case, Lemma 3.1 provides a motivation for choosing $\{\varepsilon_{top}(n)\}_{n=1}^\infty$ as a topological bandwidth selector since – while it is difficult to analyse $\varepsilon_{min}(n)$ asymptotically – at least the second summand of $\varepsilon_{top}(n)$ has the same asymptotics in $n$ as the optimal AMISE solution. Furthermore, this choice of bandwidth then corresponds to a support region $\Omega_{\varepsilon_{top}(n)}(S)$ with the correct Betti numbers (as approximated by the Vietoris-Rips complex) since $\varepsilon_{top}(n) \in [\varepsilon_{min}(n), \varepsilon_{max}(n)]$. Finally, ii) and iii) then imply that, point-wise, $\lim_{n\to\infty} \mathbb{E}[(\hat{f}_{\varepsilon_{top}(n),n}(x) - f(x))^2] = 0$ due to the results of [14, 15].

We note here that many different methods for choosing $\varepsilon(n) \in [\varepsilon_{min}(n), \varepsilon_{max}(n)]$ can be considered. If the topologically admissible interval $[\varepsilon_{min}(n), \varepsilon_{max}(n)]$ is for example determined by the constraint of having three connected components of supp $f$ as in 3(a), $\varepsilon_{max}(n)$ will increase if we shift the connected components of supp $f$ further apart. $\varepsilon_{top}(n)$ hence also increases and might not yield good $L^2$ error results for small sample sizes anymore. In that case, an estimator $\hat{\varepsilon}_{top}(n) \in [\varepsilon_{min}(n), \varepsilon_{max}(n)]$ closer to $\varepsilon_{min}(n)$ might be a better choice. To give an initial overview, we hence also display results for $\varepsilon_{min}(n), \varepsilon_{mid}(n), \varepsilon_{max}(n)$ in our experiments. Note however also that the $L^2$ error might not be the right quality measure for applications where the topological features of supp $f$ are most important – we illustrate an example of this situation in our racetrack data experiment. We will show that – in the absence of further problem-specific knowledge – $\varepsilon_{top}(n)$ does yields a good bandwidth estimate with respect to the $L^2$ error in our examples.

## 4 Experiments

**Results in 1D** We consider the probability density $f : \mathbb{R} \to \mathbb{R}$ displayed in grey in each of the graphs in Figure 3. To benchmark the performance of our topological bandwidth estimators, we then compute the AMISE-optimal bandwidth parameter $\varepsilon_{amise}$ numerically from the analytic formula for $f$ and for $K_t$, $K_u$, $K_c$ and $K_e$. Here, we include the Gaussian kernel $K_e$ for comparison purposes only.

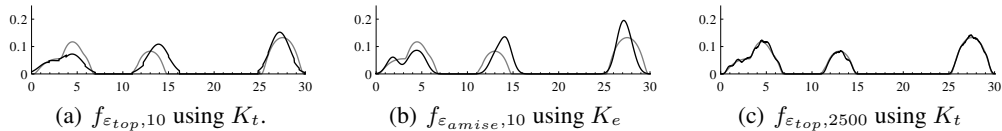

(a) $f_{\varepsilon_{top},10}$ using $K_t$.     (b) $f_{\varepsilon_{amise},10}$ using $K_e$     (c) $f_{\varepsilon_{top},2500}$ using $K_t$

Figure 3: Density $f$ (grey) and reconstructions (black) for the indicated sample size, bandwidth and kernel.

In order to topologically reconstruct $f$, we then assume *only the knowledge of some points sampled from $f$ and that $b_0(\text{supp } f) = 3$* and no further information about $f$, i.e. we assume to know a sample and that the support region of $f$ has three components. We then find $\varepsilon_{top}(n)$ by computing a topologically admissible interval $[\varepsilon_{min}(n), \varepsilon_{max}(n)]$ from the barcode corresponding to the given sample. To evaluate the quality of bandwidth parameters chosen inside $[\varepsilon_{min}(n), \varepsilon_{max}(n)]$, we then sample at various sampling sizes and compute the mean $L^2$ errors for the resulting density estimator $f_{\varepsilon,n}$ for $\varepsilon = \varepsilon_{top}, \varepsilon_{min}, \varepsilon_{max}$ and $\varepsilon_{mid} = \frac{1}{2}(\varepsilon_{max} + \varepsilon_{min})$ for each of the spherical kernels that we have described and compare our results to $\varepsilon_{amise}$. We set $\sigma^2 = \frac{1}{4}$ for $K_e$ and $K_t$. The results, summarized in Figure 4, show that $\varepsilon_{top}$ performs at a level comparable to $\varepsilon_{amise}$ in our experiments. Note here that $\varepsilon_{amise}$ can only be computed if the true density $f$ is known, while, for $\varepsilon_{top}$, we only

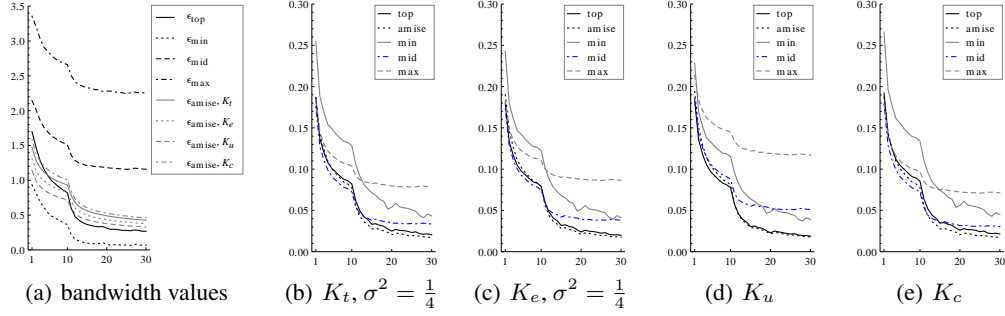

Figure 4: We generate samples from our 1D density using rejection sampling and consider sample sizes $n$ from 10 to 100 in increments of 10 (small scale) and from 250 to 5000 in increments of 250 (larger scale), resulting in 30 increasing sample sizes $n_1, \ldots, n_{30}$. In order to obtain stable results, we perform the sampling for each sampling size 1000 times (small scale), 100 times (for $250, 500, 750, 1000$) and 10 times (for $n > 1000$) respectively. We then compute the corresponding kernel density estimators $\hat{f}_{\varepsilon,n}$ and the mean $L^2$ norm of $f - \hat{f}_{\varepsilon_n,n}$. Figures (b)-(e) display these mean $L^2$ errors (vertical axis) for the indicated kernel function and bandwidth selectors. Figure (a) displays the bandwidth values (vertical axis) for the given bandwidth selectors. In all the above plots, a horizontal coordinate of $i \in \{1, \ldots, 30\}$ corresponds to a sample size of $n_i$.

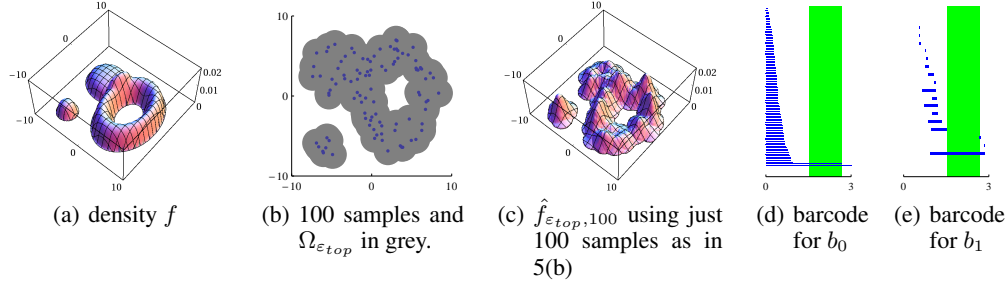

Figure 5: 2D density, samples with inferred support region $\Omega_{\varepsilon_{top}}$, topological reconstruction (using $K_t$, $\sigma^2 = \frac{1}{4}$) and barcodes with $[\varepsilon_{min}, \varepsilon_{max}]$ highlighted.

required the information that $b_0(\text{supp } f) = 3$. In our experiments (sample sizes $n \geqslant 10$), we were able to determine a valid interval $[\varepsilon_{min}(n), \varepsilon_{max}(n)]$ in all cases and did not encounter a case where the topological reconstruction was impossible.

**Results in 2D**   Here, we consider the density $f$ displayed in Figure 5(a). We chose this example to be representative for problems also arising in robotics, where the localization of a robot can be modelled as depending on a probability prior which encodes space occupied by objects by zero probability. In such scenarios, we might be able to obtain topological information about the unobstructed space $X$, such as knowing the number of components or holes in $X$. Such information could be particularly valuable in the case of deformable obstacles since their homology stays invariant under continuous deformations by homotopies. We set up the current experiment in a fashion similar to our 1D experiments, i.e. we iterate sampling from the given density for various sample sizes and compute the resulting mean $L^2$ errors to evaluate our results. As we can see from Figure 6, our results indicate that bandwidths $\varepsilon \in [\varepsilon_{min}, \varepsilon_{max}]$ yield errors comparable with the AMISE optimal bandwidth choice. While $\varepsilon_{top}$ does not perform as well as in the previous experiment, we can observe that the corresponding $L^2$ errors nonetheless follow a decreasing trend. Note also that both in 1D and 2D, $\varepsilon_{top}$ also yields good $L^2$ error results for the standard spherical Gaussian kernel here. In applications such as probabilistic motion planning, the inferred structure of supp $f$ is however of importance as well (e.g. since path-connectedness of supp $f$ is important), making a bounded support kernel a preferable choice (see also our racetrack example).

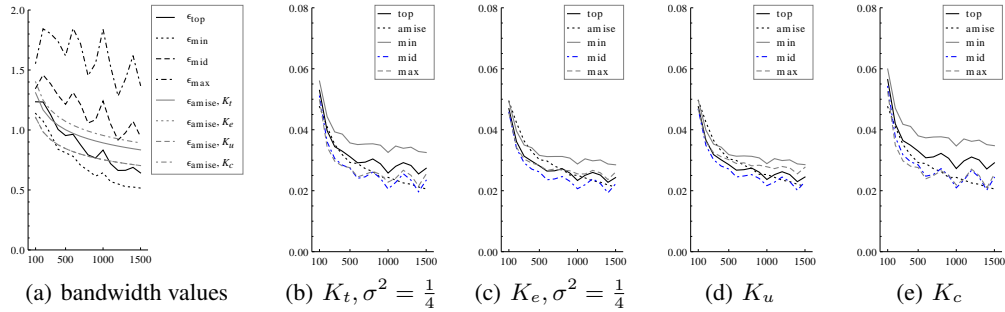

| (a) bandwidth values | (b) $K_t, \sigma^2 = \frac{1}{4}$ | (c) $K_e, \sigma^2 = \frac{1}{4}$ | (d) $K_u$ | (e) $K_c$ |

Figure 6: We generate samples from our 2D density using rejection sampling and consider sample sizes from 100 to 1500 in increments of 100. We perform sampling 10 times for each sample size and compute the corresponding kernel-based density estimator $\hat{f}_{\varepsilon,n}$ and the mean $L^2$ norm of $f - \hat{f}_{\varepsilon_n,n}$. Figures (b)-(e) display these mean $L^2$ errors (vertical axis) for the indicated sample size (horizontal axis) and kernel function. Figure (a) displays the indicated bandwidth values (vertical axis) and sample size (horizontal axis).

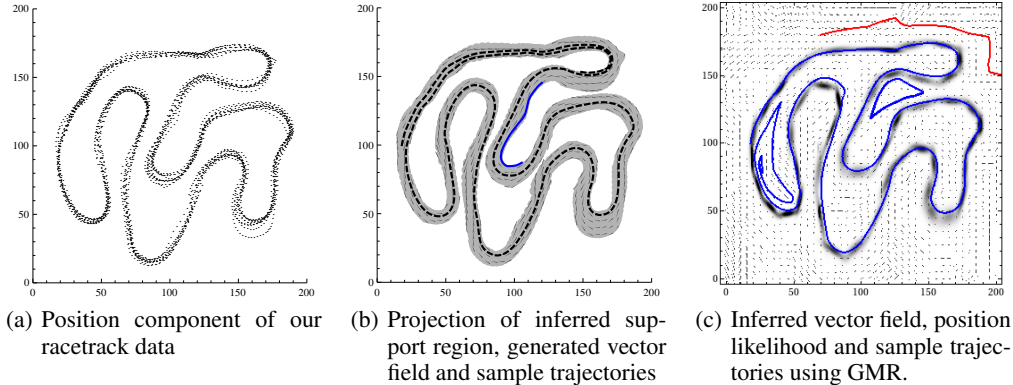

| (a) Position component of our racetrack data | (b) Projection of inferred support region, generated vector field and sample trajectories | (c) Inferred vector field, position likelihood and sample trajectories using GMR. |

Figure 7: Figure (a) shows the positions of a race car driving 10 laps around a racetrack. In (b), the results of our proposed method are displayed while Figure (c) shows the standard GMR approach. We exploit the topological information that a racetrack should be connected and 'circular' when learning the density. As can be seen, our model correctly infers the region of support as the track (grey). Using GMR, on the other hand, a non-zero probability is assigned to each location. We observe that the most probable regions are also lying over the track (black being more likely). However, when sampling new trajectories using the learned density, we can see that, whereas the trajectories using our method are confined to the track, the GMM results in undesirable trajectories.

**Application to regression**   We now consider how our framework can be applied to learn complex dynamics given a topological constraint. We consider GPS/timestamp data from 10 laps of a race car driving around a racetrack which was provided to us by [20]. For this dataset (see Figure 7(a)), we are given no information on what the boundaries of the racetrack are. One state of the art approach to modelling data like this is to employ a learning by demonstration [12] technique which is prominent especially in the context of robotics, where one attempts to learn motion patterns by observing a few demonstrations. There, one uses data points $S = \{(P_k, V_k) \in \mathbb{R}^{2n}, k = 1 \ldots n\}$, where $P_k$ describes the position and $V_k \in \mathbb{R}^n$ the associated velocity at the given position. In order to model the dynamics, one can then employ a Gaussian mixture model [12] in $\mathbb{R}^{2n}$ to learn a probability density $\hat{f}$ for the dataset (usually using the EM-algorithm). To every position $x \in \mathbb{R}^n$, one can then associate the velocity vector given by $\mathbb{E}(V|P = x)$ with respect to the learned density $\hat{f}$ – this uses the idea of Gaussian mixture regression (GMR). The resulting vector field can then be numerically integrated to yield new trajectories. Since $\mathbb{E}(V|P = x)$ for a Gaussian mixture model can be computed easily, this method can be applied even in high-dimensional spaces. While it can be considered as a strength of the GMR approach that it is able to infer – from just a few examples –

a vector field that is non-zero on a dense subset of $\mathbb{R}^n$, this can also be problematic since geometric and topological constraints are not naturally part of this approach and we cannot easily encode the fact that the vector-field should be non-zero only on the racetrack.

From our GPS/timestamp data, we now compute velocity vectors for each data-point and embed the data in the manner just described in $\mathbb{R}^4$. We then experimented with the software [21] to model our racetrack data with a mixture of a varying number of Gaussians. While the model brakes down completely for a low number of Gaussians, some interesting behaviour can be observed in the case of a mixture model with 50 Gaussians displayed in Figure 7(c). We display the resulting velocity vector field together with several newly synthesized trajectories. We observe both an undesired periodic trajectory as well as a trajectory that almost completely traverses the racetrack before converging towards an attractor. The likelihood of a given position is additionally displayed in 7(c) with black being the most likely. While the most likely positions do occur over the racetrack, the mixture model does not provide a natural way of determining where the boundaries of the track should lie. The topmost trajectory in 7(c), for example, starts at a highly unlikely position.

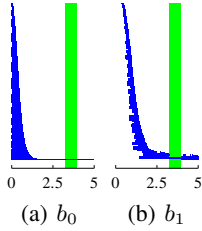

(a) $b_0$  (b) $b_1$

Figure 8: Barcodes in dimension zero (a) and one (b) and shaded $[\varepsilon_{min}, \varepsilon_{max}]$ interval for our racetrack.

Let us now consider how we can apply the density estimation techniques we have described in this paper in this case. Given that we know that the racetrack is a closed curve, we assume that the data should be modelled by a probability density $f : \mathbb{R}^4 \to \mathbb{R}$ whose support region $\Omega$ has a single component $(b_0(\Omega) = 1)$ and $\Omega$ should topologically be a circle $(b_1(\Omega) = 1)$. In order for the velocities of differing laps around the track not to lie too far apart , and so that the topology of the racetrack dominates in $\mathbb{R}^4$, we rescale the velocity components of our data to lie inside the interval $[-0.6, 0.6]$. Figure 8 displays the barcode for our data. Using our procedure, we compute that $[\varepsilon_{min}, \varepsilon_{max}] \cong [3.25, 3.97]$ is the bandwidth interval for which the topological constraints that we just defined are satisfied. Using the kernel $K_t$ with $\sigma^2 = \frac{1}{4}$ and the corresponding density estimator $\hat{f}_{\varepsilon top}$, we obtain $\Omega_{\varepsilon top} \subset \mathbb{R}^4$ with the correct topological properties. Figure 7(b) displays the projection of $\Omega_{\varepsilon top}$ onto $\mathbb{R}^2$. As a next step, we suggest to follow the idea of the GMR approach to compute the posterior expectation $\mathbb{E}(V | P = x)$, but this time for our density $\hat{f}_{\varepsilon top}$. It follows from the definition of our kernel-based estimator that, for $x$ such that $(x, y) \in \Omega_{\varepsilon top}$ for some $y \in \mathbb{R}^n$, we have $\mathbb{E}(V | P = x) = \frac{\sum_{i=1}^n Y_i \int K_t\left(\frac{x - X_i}{\varepsilon_{top}}, z\right) \mathrm{d}z}{\sum_{i=1}^n \int K_t\left(\frac{x - X_i}{\varepsilon_{top}}, z\right) \mathrm{d}z}$. While we were not able to find a reference for the use or computation of these marginals for spherical truncated Gaussians, a reasonably simple calculation shows that these can in fact be computed analytically in arbitrary dimension:

**Lemma 4.1.** *Consider $d, k \in \mathbb{N}$, $d > k$ and $x \in \mathbb{R}^k$. Let $K_t : \mathbb{R}^d \to \mathbb{R}$ denote the spherical truncated Gaussian with parameter $\sigma^2 > 0$. Then*

$$\int_{\mathbb{R}^{d-k}} K_t(x, y)\mathrm{d}y = \frac{1}{(2\pi\sigma^2)^{k/2}} \frac{P\left(\frac{d-k}{2}, \frac{1-\|x\|^2}{2\sigma^2}\right)}{P\left(\frac{d}{2}, \frac{1}{2\sigma^2}\right)} e^{-\frac{\|x\|^2}{2\sigma^2}}$$

*for $\|x\| \leqslant 1$ and 0 otherwise. Here, $P(a, b) = 1 - \frac{\Gamma(a,b)}{\Gamma(a)}$ denotes the normalized Gamma P function.*

For every point in the projection of $\Omega_{\varepsilon top}$ onto the position coordinates, we can hence compute a velocity $\mathbb{E}(V | P = x)$ to generate new motion trajectories. For points outside the support region, we postulate zero velocity. Figure 7(c) displays the resulting vector-field and a few sample trajectories. As we can see, these follow the trajectory of the data points in Figure 7(a) very well. At the same time, the displayed support region looks like a sensible choice for the position of the racetrack.

## 5   Conclusion

In this paper, we have presented a novel method for learning density models with bounded support. The proposed topological bandwidth selection approach allows to incorporate topological constraints within a probabilistic modelling framework by combining algebraic-topological information obtained in terms of persistent homology with tools from kernel-based density estimation. We have provided a first thorough evaluation of the $L^2$ errors for synthetic data and have exemplified the practical use of our approach through application in a learning by demonstration scenario.

# References

[1] D. A. Reynolds, T. F. Quatieri, and R. B. Dunn, "Speaker verification using adapted Gaussian mixture models," *Digital Signal Processing*, vol. 10, no. 1–3, pp. 19–41, 2000.

[2] C. E. Rasmussen and C. Williams, *Gaussian Processes for Machine Learning*. MIT Press, 2006.

[3] D. A. Cohn, Z. Ghahramani, and M. I. Jordan, "Active learning with statistical models," *Journal of Artificial Intelligence Research*, no. 4, pp. 129–145, 1996.

[4] S. Calinon and A. Billard, "Incremental learning of gestures by imitation in a humanoid robot," in *ACM/IEEE International Conference on Human-Robot Interaction*, pp. 255–262, 2007.

[5] D.-S. Lee, "Effective Gaussian mixture learning for video background subtraction," *PAMI*, vol. 27, no. 5, pp. 827–832, 2005.

[6] M. P. Wand and M. C. Jones, *Kernel Smoothing*, vol. 60 of *Monographs on Statistics and Applied Probability*. Chapman and Hall/CRC, 1995.

[7] B. A. Turlach, "Bandwidth selection in kernel density estimation: A review," in *CORE and Institut de Statistique*, pp. 23–493, 1993.

[8] L. El Ghaoui and G. Calafiore, "Robust filtering for discrete-time systems with bounded noise and parametric uncertainty," *IEEE Transactions on Automatic Control*, vol. 46, no. 7, pp. 1084–1089, 2001.

[9] Y. C. Eldar, A. Ben-Tal, and A. Nemirovski, "Linear minimax regret estimation of deterministic parameters with bounded data uncertainties," *IEEE Transactions on Signal Processing*, vol. 52, no. 8, pp. 2177–2188, 2008.

[10] G. Carlsson, "Topology and data," *Bull. Amer. Math. Soc. (N.S.)*, vol. 46, no. 2, pp. 255–308, 2009.

[11] P. Niyogi, S. Smale, and S. Weinberger, "A topological view of unsupervised learning from noisy data," *SIAM Journal of Computing*, vol. 40, no. 3, pp. 646–663, 2011.

[12] S. M. Khansari-Zadeh and A. Billard, "Learning stable non-linear dynamical systems with Gaussian mixture models," *IEEE Transaction on Robotics*, vol. 27, no. 5, pp. 943–957, 2011.

[13] M. Rosenblatt, "Remarks on some nonparametric estimates of a density function," *The Annals of Mathematical Statistics*, vol. 27, no. 3, pp. 832–837, 1956.

[14] E. Parzen, "On estimation of a probability density function and mode," *Annals of Mathematical Statistics*, vol. 33, pp. 1065–1076, 1962.

[15] T. Cacoullos, "Estimation of a multivariate density," *Annals of the Institute of Statistical Mathematics*, vol. 18, pp. 179–189, 1966.

[16] H. Edelsbrunner, D. Letscher, and A. Zomorodian, "Topological persistence and simplification," *Discrete Comput. Geom.*, vol. 28, no. 4, pp. 511–533, 2002.

[17] A. Hatcher, *Algebraic Topology*. Cambridge University Press, 2002.

[18] P. Niyogi, S. Smale, and S. Weinberger, "Finding the homology of submanifolds with high confidence from random samples," *Discrete Comput. Geom.*, vol. 39, no. 1-3, pp. 419–441, 2008.

[19] A. Tausz, M. Vejdemo-Johansson, and H. Adams, "JavaPlex: A software package for computing persistent topological invariants." Software, 2011.

[20] KTH Racing, Formula Student Team, KTH Royal Institute of Technology, Stockholm, Sweden.

[21] A. Billard, "GMM/GMR 2.0." Software.

